# Efficient sparse coding algorithms

**Honglak Lee**   **Alexis Battle**   **Rajat Raina**   **Andrew Y. Ng**
Computer Science Department
Stanford University
Stanford, CA 94305

## Abstract

Sparse coding provides a class of algorithms for finding succinct representations of stimuli; given only unlabeled input data, it discovers basis functions that capture higher-level features in the data. However, finding sparse codes remains a very difficult computational problem. In this paper, we present efficient sparse coding algorithms that are based on iteratively solving two convex optimization problems: an $L_1$-regularized least squares problem and an $L_2$-constrained least squares problem. We propose novel algorithms to solve both of these optimization problems. Our algorithms result in a significant speedup for sparse coding, allowing us to learn larger sparse codes than possible with previously described algorithms. We apply these algorithms to natural images and demonstrate that the inferred sparse codes exhibit end-stopping and non-classical receptive field surround suppression and, therefore, may provide a partial explanation for these two phenomena in V1 neurons.

## 1 Introduction

Sparse coding provides a class of algorithms for finding succinct representations of stimuli; given only unlabeled input data, it learns basis functions that capture higher-level features in the data. When a sparse coding algorithm is applied to natural images, the learned bases resemble the receptive fields of neurons in the visual cortex [1, 2]; moreover, sparse coding produces localized bases when applied to other natural stimuli such as speech and video [3, 4]. Unlike some other unsupervised learning techniques such as PCA, sparse coding can be applied to learning overcomplete basis sets, in which the number of bases is greater than the input dimension. Sparse coding can also model inhibition between the bases by sparsifying their activations. Similar properties have been observed in biological neurons, thus making sparse coding a plausible model of the visual cortex [2, 5].

Despite the rich promise of sparse coding models, we believe that their development has been hampered by their expensive computational cost. In particular, learning large, highly overcomplete representations has been extremely expensive. In this paper, we develop a class of efficient sparse coding algorithms that are based on alternating optimization over two subsets of the variables. The optimization problems over each of the subsets of variables are convex; in particular, the optimization over the first subset is an $L_1$-regularized least squares problem; the optimization over the second subset of variables is an $L_2$-constrained least squares problem. We describe each algorithm and empirically analyze their performance. Our method allows us to efficiently learn large overcomplete bases from natural images. We demonstrate that the resulting learned bases exhibit (i) end-stopping [6] and (ii) modulation by stimuli outside the classical receptive field (nCRF surround suppression) [7]. Thus, sparse coding may also provide a partial explanation for these phenomena in V1 neurons. Further, in related work [8], we show that the learned succinct representation captures higher-level features that can then be applied to supervised classification tasks.

## 2 Preliminaries

The goal of sparse coding is to represent input vectors approximately as a weighted linear combination of a small number of (unknown) "basis vectors." These basis vectors thus capture high-level patterns in the input data. Concretely, each input vector $\vec{\xi} \in \mathbb{R}^k$ is succinctly represented using basis vectors $\vec{b}_1, \ldots, \vec{b}_n \in \mathbb{R}^k$ and a sparse vector of weights or "coefficients" $\vec{s} \in \mathbb{R}^n$ such that $\vec{\xi} \approx \sum_j \vec{b}_j s_j$. The basis set can be overcomplete ($n > k$), and can thus capture a large number of patterns in the input data.

Sparse coding is a method for discovering good basis vectors automatically using only unlabeled data. The standard generative model assumes that the reconstruction error $\vec{\xi} - \sum_j \vec{b}_j s_j$ is distributed as a zero-mean Gaussian distribution with covariance $\sigma^2 I$. To favor sparse coefficients, the prior distribution for each coefficient $s_j$ is defined as: $P(s_j) \propto \exp(-\beta\phi(s_j))$, where $\phi(\cdot)$ is a sparsity function and $\beta$ is a constant. For example, we can use one of the following:

$$\phi(s_j) = \begin{cases} \|s_j\|_1 & \text{(L}_1 \text{ penalty function)} \\ (s_j^2 + \epsilon)^{\frac{1}{2}} & \text{(epsilonL}_1 \text{ penalty function)} \\ \log(1 + s_j^2) & \text{(log penalty function).} \end{cases} \qquad (1)$$

In this paper, we will use the $L_1$ penalty unless otherwise mentioned; $L_1$ regularization is known to produce sparse coefficients and can be robust to irrelevant features [9].

Consider a training set of $m$ input vectors $\vec{\xi}^{(1)}, ..., \vec{\xi}^{(m)}$, and their (unknown) corresponding coefficients $\vec{s}^{(1)}, ..., \vec{s}^{(m)}$. The maximum a posteriori estimate of the bases and coefficients, assuming a uniform prior on the bases, is the solution to the following optimization problem:[1]

$$\text{minimize}_{\{\vec{b}_j\},\{\vec{s}^{(i)}\}} \quad \sum_{i=1}^{m} \frac{1}{2\sigma^2} \|\vec{\xi}^{(i)} - \sum_{j=1}^{n} \vec{b}_j s_j^{(i)}\|^2 + \beta \sum_{i=1}^{m} \sum_{j=1}^{n} \phi(s_j^{(i)}) \qquad (2)$$

$$\text{subject to} \qquad \|\vec{b}_j\|^2 \leq c, \ \forall j = 1, ..., n.$$

This problem can be written more concisely in matrix form: let $X \in \mathbb{R}^{k \times m}$ be the input matrix (each column is an input vector), let $B \in \mathbb{R}^{k \times n}$ be the basis matrix (each column is a basis vector), and let $S \in \mathbb{R}^{n \times m}$ be the coefficient matrix (each column is a coefficient vector). Then, the optimization problem above can be written as:

$$\text{minimize}_{B,S} \quad \frac{1}{2\sigma^2} \|X - BS\|_F^2 + \beta \sum_{i,j} \phi(S_{i,j}) \qquad (3)$$

$$\text{subject to} \qquad \sum_i B_{i,j}^2 \leq c, \ \forall j = 1, ..., n.$$

Assuming the use of either $L_1$ penalty or epsilonL$_1$ penalty as the sparsity function, the optimization problem is convex in $B$ (while holding $S$ fixed) and convex in $S$ (while holding $B$ fixed),[2] but not convex in both simultaneously. In this paper, we iteratively optimize the above objective by alternatingly optimizing with respect to $B$ (bases) and $S$ (coefficients) while holding the other fixed.

For learning the bases $B$, the optimization problem is a least squares problem with quadratic constraints. There are several approaches to solving this problem, such as generic convex optimization solvers (e.g., QCQP solver) as well as gradient descent using iterative projections [10]. However, generic convex optimization solvers are too slow to be applicable to this problem, and gradient descent using iterative projections often shows slow convergence. In this paper, we derive and solve the Lagrange dual, and show that this approach is much more efficient than gradient-based methods.

For learning the coefficients $S$, the optimization problem is equivalent to a regularized least squares problem. For many differentiable sparsity functions, we can use gradient-based methods (e.g., conjugate gradient). However, for the $L_1$ sparsity function, the objective is not continuously differentiable and the most straightforward gradient-based methods are difficult to apply. In this case, the following approaches have been used: generic QP solvers (e.g., CVX), Chen et al.'s interior point method [11], a modification of least angle regression (LARS) [12], or grafting [13]. In this paper, we present a new algorithm for solving the $L_1$-regularized least squares problem and show that it is more efficient for learning sparse coding bases.

## 3 $L_1$-regularized least squares: The feature-sign search algorithm

Consider solving the optimization problem (2) with an $L_1$ penalty over the coefficients $\{s_j^{(i)}\}$ while keeping the bases fixed. This problem can be solved by optimizing over each $\vec{s}^{(i)}$ individually:

$$\text{minimize}_{\vec{s}^{(i)}} \|\vec{\xi}^{(i)} - \sum_j \vec{b}_j s_j^{(i)}\|^2 + (2\sigma^2\beta) \sum_j |s_j^{(i)}|. \qquad (4)$$

Notice now that if we know the signs (positive, zero, or negative) of the $s_j^{(i)}$'s at the optimal value, we can replace each of the terms $|s_j^{(i)}|$ with either $s_j^{(i)}$ (if $s_j^{(i)} > 0$), $-s_j^{(i)}$ (if $s_j^{(i)} < 0$), or 0 (if

$s_j^{(i)} = 0$). Considering only nonzero coefficients, this reduces (4) to a standard, unconstrained quadratic optimization problem (QP), which can be solved analytically and efficiently. Our algorithm, therefore, tries to search for, or "guess," the signs of the coefficients $s_j^{(i)}$; given any such guess, we can efficiently solve the resulting unconstrained QP. Further, the algorithm systematically refines the guess if it turns out to be initially incorrect.

To simplify notation, we present the algorithm for the following equivalent optimization problem:

$$\text{minimize}_x f(x) \equiv \|y - Ax\|^2 + \gamma\|x\|_1, \tag{5}$$

where $\gamma$ is a constant. The feature-sign search algorithm is shown in Algorithm 1. It maintains an *active set* of potentially nonzero coefficients and their corresponding signs—all other coefficients must be zero—and systematically searches for the optimal active set and coefficient signs. The algorithm proceeds in a series of "feature-sign steps": on each step, it is given a current guess for the active set and the signs, and it computes the analytical solution $\hat{x}_{new}$ to the resulting unconstrained QP; it then updates the solution, the active set and the signs using an efficient discrete line search between the current solution and $\hat{x}_{new}$ (details in Algorithm 1).[3] We will show that each such step reduces the objective $f(x)$, and that the overall algorithm always converges to the optimal solution.

---

**Algorithm 1** Feature-sign search algorithm

---

1 Initialize $x := \vec{0}, \theta := \vec{0}$, and *active set* $:= \{\}$, where $\theta_i \in \{-1, 0, 1\}$ denotes $\text{sign}(x_i)$.
2 From zero coefficients of $x$, select $i = \arg\max_i \left| \frac{\partial\|y - Ax\|^2}{\partial x_i} \right|$.
    Activate $x_i$ (add $i$ to the *active set*) only if it locally improves the objective, namely:
        If $\frac{\partial\|y - Ax\|^2}{\partial x_i} > \gamma$, then set $\theta_i := -1$, *active set* $:= \{i\} \cup$ *active set*.
        If $\frac{\partial\|y - Ax\|^2}{\partial x_i} < -\gamma$, then set $\theta_i := 1$, *active set* $:= \{i\} \cup$ *active set*.
3 Feature-sign step:
    Let $\hat{A}$ be a submatrix of $A$ that contains only the columns corresponding to the *active set*.
    Let $\hat{x}$ and $\hat{\theta}$ be subvectors of $x$ and $\theta$ corresponding to the *active set*.
    Compute the analytical solution to the resulting unconstrained QP ($\text{minimize}_{\hat{x}} \|y - \hat{A}\hat{x}\|^2 + \gamma\hat{\theta}^\top\hat{x}$):
        $\hat{x}_{new} := (\hat{A}^\top\hat{A})^{-1}(\hat{A}^\top y - \gamma\hat{\theta}/2)$,
    Perform a discrete line search on the closed line segment from $\hat{x}$ to $\hat{x}_{new}$:
        Check the objective value at $\hat{x}_{new}$ and all points where any coefficient changes sign.
        Update $\hat{x}$ (and the corresponding entries in $x$) to the point with the lowest objective value.
    Remove zero coefficients of $\hat{x}$ from the *active set* and update $\theta := \text{sign}(x)$.
4 Check the optimality conditions:
    (a) Optimality condition for nonzero coefficients: $\frac{\partial\|y - Ax\|^2}{\partial x_j} + \gamma\,\text{sign}(x_j) = 0, \forall x_j \neq 0$
        If condition (a) is not satisfied, go to Step 3 (without any new activation); else check condition (b).
    (b) Optimality condition for zero coefficients: $\left| \frac{\partial\|y - Ax\|^2}{\partial x_j} \right| \leq \gamma, \forall x_j = 0$
        If condition (b) is not satisfied, go to Step 2; otherwise return $x$ as the solution.

---

To sketch the proof of convergence, let a coefficient vector $x$ be called *consistent* with a given active set and sign vector $\theta$ if the following two conditions hold for all $i$: (i) If $i$ is in the active set, then $\text{sign}(x_i) = \theta_i$, and, (ii) If $i$ is not in the active set, then $x_i = 0$.

**Lemma 3.1.** *Consider optimization problem (5) augmented with the additional constraint that $x$ is consistent with a given active set and sign vector. Then, if the current coefficients $x_c$ are consistent with the active set and sign vector, but are not optimal for the augmented problem at the start of Step 3, the feature-sign step is guaranteed to strictly reduce the objective.*

*Proof sketch.* Let $\hat{x}_c$ be the subvector of $x_c$ corresponding to coefficients in the given active set. In Step 3, consider a smooth quadratic function $\tilde{f}(\hat{x}) \equiv \|y - \hat{A}\hat{x}\|^2 + \gamma\hat{\theta}^\top\hat{x}$. Since $\hat{x}_c$ is not an optimal point of $\tilde{f}$, we have $\tilde{f}(\hat{x}_{new}) < \tilde{f}(\hat{x}_c)$. Now consider the two possible cases: (i) if $\hat{x}_{new}$ is consistent with the given active set and sign vector, updating $\hat{x} := \hat{x}_{new}$ strictly decreases the objective; (ii) if $\hat{x}_{new}$ is not consistent with the given active set and sign vector, let $\hat{x}_d$ be the first zero-crossing point (where any coefficient changes its sign) on a line segment from $\hat{x}_c$ to $\hat{x}_{new}$, then clearly $\hat{x}_c \neq \hat{x}_d$,

and $\tilde{f}(\hat{x}_d) < \tilde{f}(\hat{x}_c)$ by convexity of $\tilde{f}$, thus we finally have $f(\hat{x}_d) = \tilde{f}(\hat{x}_d) < \tilde{f}(\hat{x}_c) = f(\hat{x}_c)$.[4] Therefore, the discrete line search described in Step 3 ensures a decrease in the objective value. $\square$

**Lemma 3.2.** *Consider optimization problem (5) augmented with the additional constraint that $x$ is consistent with a given active set and sign vector. If the coefficients $x_c$ at the start of Step 2 are optimal for the augmented problem, but are not optimal for problem (5), the feature-sign step is guaranteed to strictly reduce the objective.*

*Proof sketch.* Since $x_c$ is optimal for the augmented problem, it satisfies optimality condition (a), but not (b); thus, in Step 2, there is some $i$, such that $\left| \frac{\partial \|y - Ax\|^2}{\partial x_i} \right| > \gamma$; this $i$-th coefficient is activated, and $i$ is added to the active set. In Step 3, consider the smooth quadratic function $\tilde{f}(\hat{x}) \equiv \|y - \hat{A}\hat{x}\|^2 + \gamma \hat{\theta}^\top \hat{x}$. Observe that (i) since a Taylor expansion of $\tilde{f}$ around $\hat{x} = \hat{x}_c$ has a first order term in $x_i$ only (using condition 4(a) for the other coefficients), we have that any direction that locally decreases $\tilde{f}(\hat{x})$ must be consistent with the sign of the activated $x_i$, and, (ii) since $\hat{x}_c$ is not an optimal point of $\tilde{f}(\hat{x})$, $\tilde{f}(\hat{x})$ must decrease locally near $\hat{x} = \hat{x}_c$ along the direction from $\hat{x}_c$ to $\hat{x}_{new}$. From (i) and (ii), the line search direction $\hat{x}_c$ to $\hat{x}_{new}$ must be consistent with the sign of the activated $x_i$. Finally, since $\tilde{f}(\hat{x}) = f(\hat{x})$ when $\hat{x}$ is consistent with the active set, either $\hat{x}_{new}$ is consistent, or the first zero-crossing from $\hat{x}_c$ to $\hat{x}_{new}$ has a lower objective value (similar argument to Lemma 3.1). $\square$

**Theorem 3.3.** *The feature-sign search algorithm converges to a global optimum of the optimization problem (5) in a finite number of steps.*

*Proof sketch.* From the above lemmas, it follows that the feature-sign steps always strictly reduce the objective $f(x)$. At the start of Step 2, $x$ either satisfies optimality condition 4(a) or is $\vec{0}$; in either case, $x$ is consistent with the current active set and sign vector, and must be optimal for the augmented problem described in the above lemmas. Since the number of all possible active sets and coefficient signs is finite, and since no pair can be repeated (because the objective value is strictly decreasing), the outer loop of Steps 2–4(b) cannot repeat indefinitely. Now, it suffices to show that a finite number of steps is needed to reach Step 4(b) from Step 2. This is true because the inner loop of Steps 3–4(a) always results in either an exit to Step 4(b) or a decrease in the size of the active set. $\square$

Note that initialization with arbitrary starting points requires a small modification: after initializing $\theta$ and the active set with a given initial solution, we need to start with Step 3 instead of Step 1.[5] When the initial solution is near the optimal solution, feature-sign search can often obtain the optimal solution more quickly than when starting from $\vec{0}$.

## 4    Learning bases using the Lagrange dual

In this subsection, we present a method for solving optimization problem (3) over bases $B$ given fixed coefficients $S$. This reduces to the following problem:

$$\text{minimize} \qquad \|X - BS\|_{\mathrm{F}}^2 \tag{6}$$
$$\text{subject to} \quad \sum_{i=1}^{k} B_{i,j}^2 \leq c, \forall j = 1, ..., n.$$

This is a least squares problem with quadratic constraints. In general, this constrained optimization problem can be solved using gradient descent with iterative projection [10]. However, it can be much more efficiently solved using a Lagrange dual. First, consider the Lagrangian:

$$\mathcal{L}(B, \vec{\lambda}) \;\; = \;\; \text{trace}\left((X - BS)^\top (X - BS)\right) + \sum_{j=1}^{n} \lambda_j \left(\sum_{i=1}^{k} B_{i,j}^2 - c\right), \tag{7}$$

where each $\lambda_j \geq 0$ is a dual variable. Minimizing over $B$ analytically, we obtain the Lagrange dual:

$$\mathcal{D}(\vec{\lambda}) = \min_B \mathcal{L}(B, \vec{\lambda}) \;\; = \;\; \text{trace}\left(X^\top X - XS^\top (SS^\top + \Lambda)^{-1}(XS^\top)^\top - c\,\Lambda\right), \tag{8}$$

where $\Lambda = \text{diag}(\vec{\lambda})$. The gradient and Hessian of $\mathcal{D}(\vec{\lambda})$ are computed as follows:

$$\frac{\partial \mathcal{D}(\vec{\lambda})}{\partial \lambda_i} \;\; = \;\; \|XS^\top (SS^\top + \Lambda)^{-1} e_i\|^2 - c, \tag{9}$$

$$\frac{\partial^2 \mathcal{D}(\vec{\lambda})}{\partial \lambda_i \partial \lambda_j} \;\; = \;\; -2\left((SS^\top + \Lambda)^{-1}(XS^\top)^\top XS^\top (SS^\top + \Lambda)^{-1}\right)_{i,j} \left((SS^\top + \Lambda)^{-1}\right)_{i,j}, \tag{10}$$

| | natural image 196×512 | speech 500×200 | stereo 288×400 | video 512×200 |
|---|---|---|---|---|
| Feature-sign | 2.16 (0) | 0.58 (0) | 1.72 (0) | 0.83 (0) |
| LARS | 3.62 (0) | 1.28 (0) | 4.02 (0) | 1.98 (0) |
| Grafting | 13.39 (7e-4) | 4.69 (4e-6) | 11.12 (5e-4) | 5.88 (2e-4) |
| Chen et al.'s | 88.61 (8e-5) | 47.49 (8e-5) | 66.62 (3e-4) | 47.00 (2e-4) |
| QP solver (CVX) | 387.90 (4e-9) | 1,108.71 (1e-8) | 538.72 (7e-9) | 1,219.80 (1e-8) |

Table 1: The running time in seconds (and the relative error in parentheses) for coefficient learning algorithms applied to different natural stimulus datasets. For each dataset, the input dimension $k$ and the number of bases $n$ are specified as $k \times n$. The relative error for an algorithm was defined as $(f_{obj} - f^*)/f^*$, where $f_{obj}$ is the final objective value attained by that algorithm, and $f^*$ is the best objective value attained among all the algorithms.

where $e_i \in \mathbb{R}^n$ is the $i$-th unit vector. Now, we can optimize the Lagrange dual (8) using Newton's method or conjugate gradient. After maximizing $\mathcal{D}(\vec{\lambda})$, we obtain the optimal bases $B$ as follows:

$$B^\top = (SS^\top + \Lambda)^{-1}(XS^\top)^\top. \tag{11}$$

The advantage of solving the dual is that it uses significantly fewer optimization variables than the primal. For example, optimizing $B \in \mathbb{R}^{1,000 \times 1,000}$ requires only 1,000 dual variables. Note that the dual formulation is independent of the sparsity function (e.g., $L_1$, epsilon$L_1$, or other sparsity function), and can be extended to other similar models such as "topographic" cells [14].[6]

## 5 Experimental results

### 5.1 The feature-sign search algorithm

We evaluated the performance of our algorithms on four natural stimulus datasets: natural images, speech, stereo images, and natural image videos. All experiments were conducted on a Linux machine with AMD Opteron 2GHz CPU and 2GB RAM.

First, we evaluated the feature-sign search algorithm for learning coefficients with the $L_1$ sparsity function. We compared the running time and accuracy to previous state-of-the-art algorithms: a generic QP solver,[7] a modified version of LARS [12] with early stopping,[8] grafting [13], and Chen et al.'s interior point method [11];[9] all the algorithms were implemented in MATLAB. For each dataset, we used a test set of 100 input vectors and measured the running time[10] and the objective function at convergence. Table 1 shows both the running time and accuracy (measured by the relative error in the final objective value) of different coefficient learning algorithms. Over all datasets, feature-sign search achieved the best objective values as well as the shortest running times. Feature-sign search and modified LARS produced more accurate solutions than the other methods.[11] Feature-sign search was an order of magnitude faster than both Chen et al.'s algorithm and the generic QP solver, and it was also significantly faster than modified LARS and grafting. Moreover, feature-sign search has the crucial advantage that it can be initialized with arbitrary starting coefficients (unlike LARS); we will demonstrate that feature-sign search leads to even further speedup over LARS when applied to iterative coefficient learning.

### 5.2 Total time for learning bases

The Lagrange dual method for one basis learning iteration was much faster than gradient descent with iterative projections, and we omit discussion of those results due to space constraints. Below, we directly present results for the overall time taken by sparse coding for learning bases from natural stimulus datasets.

|  | $L_1$ sparsity function | | | |
| --- | --- | --- | --- | --- |
| Coeff. / Basis learning | natural image | speech | stereo | video |
| Feature-sign / LagDual | 260.0 | 248.2 | 438.2 | 186.6 |
| Feature-sign / GradDesc | 1,093.9 | 1,280.3 | 950.6 | 933.2 |
| LARS / LagDual | 666.7 | 1,697.7 | 1,342.7 | 1,254.6 |
| LARS / GradDesc | 13,085.1 | 17,219.0 | 12,174.6 | 11,022.8 |
| Grafting / LagDual | 720.5 | 1,025.5 | 3,006.0 | 1,340.5 |
| Grafting / GradDesc | 2,767.9 | 8,670.8 | 6,203.3 | 3,681.9 |
|  | $\text{epsilon}L_1$ sparsity function | | | |
| Coeff. / Basis learning | natural image | speech | stereo | video |
| ConjGrad / LagDual | 1,286.6 | 544.4 | 1,942.4 | 1,461.9 |
| ConjGrad / GradDesc | 5,047.3 | 11,939.5 | 3,435.1 | 2,479.2 |

Table 2: The running time (in seconds) for different algorithm combinations using different sparsity functions.

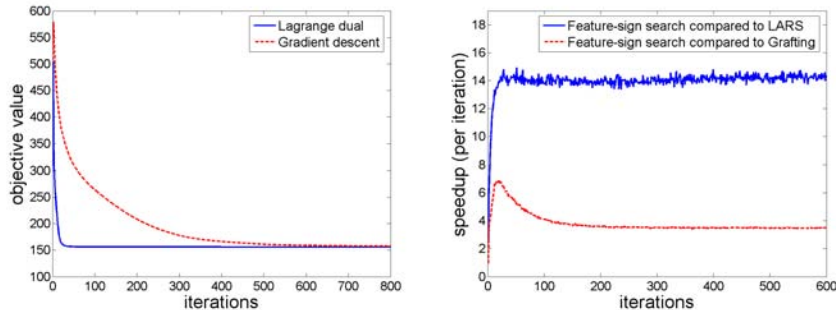

Figure 1: Demonstration of speedup. Left: Comparison of convergence between the Lagrange dual method and gradient descent for learning bases. Right: The running time per iteration for modified LARS and grafting as a multiple of the running time per iteration for feature-sign search.

We evaluated different combinations of coefficient learning and basis learning algorithms: the fastest coefficient learning methods from our experiments (feature-sign search, modified LARS and grafting for the $L_1$ sparsity function, and conjugate gradient for the $\text{epsilon}L_1$ sparsity function) and the state-of-the-art basis learning methods (gradient descent with iterative projection and the Lagrange dual formulation). We used a training set of 1,000 input vectors for each of the four natural stimulus datasets. We initialized the bases randomly and ran each algorithm combination (by alternatingly optimizing the coefficients and the bases) until convergence.[12]

Table 2 shows the running times for different algorithm combinations. First, we observe that the Lagrange dual method significantly outperformed gradient descent with iterative projections for both $L_1$ and $\text{epsilon}L_1$ sparsity; a typical convergence pattern is shown in Figure 1 (left). Second, we observe that, for $L_1$ sparsity, feature-sign search significantly outperformed both modified LARS and grafting.[13] Figure 1 (right) shows the running time per iteration for modified LARS and grafting as a multiple of that for feature-sign search (using the same gradient descent algorithm for basis learning), demonstrating significant efficiency gains at later iterations; note that feature-sign search (and grafting) can be initialized with the coefficients obtained in the previous iteration, whereas modified LARS cannot. This result demonstrates that feature-sign search is particularly efficient for iterative optimization, such as learning sparse coding bases.

### 5.3 Learning highly overcomplete natural image bases

Using our efficient algorithms, we were able to learn highly overcomplete bases of natural images as shown in Figure 2. For example, we were able to learn a set of 1,024 bases (each $14\times14$ pixels)

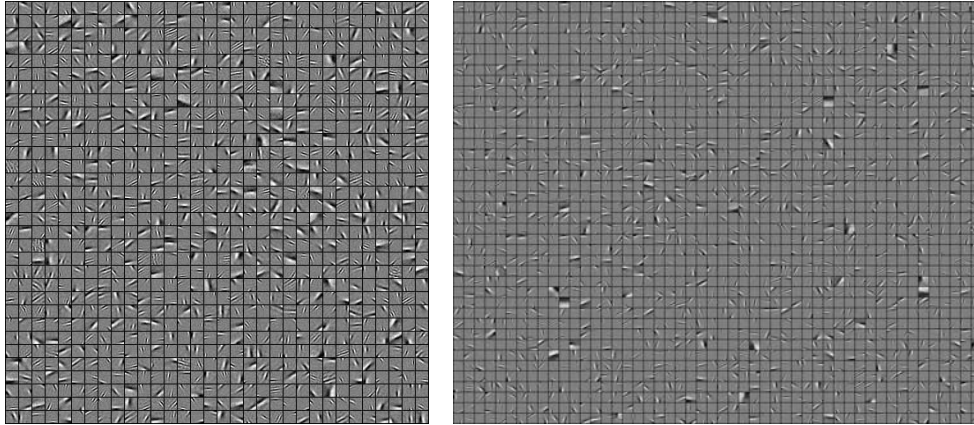

Figure 2: Learned overcomplete natural image bases. Left: 1,024 bases (each 14×14 pixels). Right: 2,000 bases (each 20×20 pixels).

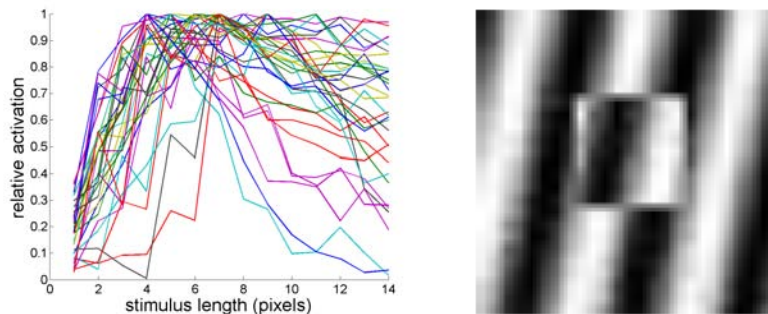

Figure 3: Left: End-stopping test for 14×14 sized 1,024 bases. Each line in the graph shows the coefficients for a basis for different length bars. Right: Sample input image for nCRF effect.

in about 2 hours and a set of 2,000 bases (each 20×20 pixels) in about 10 hours.[14] In contrast, the gradient descent method for basis learning did not result in any reasonable bases even after running for 24 hours. Further, summary statistics of our learned bases, obtained by fitting the Gabor function parameters to each basis, qualitatively agree with previously reported statistics [15].

## 5.4 Replicating complex neuroscience phenomena

Several complex phenomena of V1 neural responses are not well explained by simple linear models (in which the response is a linear function of the input). For instance, many visual neurons display "end-stopping," in which the neuron's response to a bar image of optimal orientation and placement is actually suppressed as the bar length exceeds an optimal length [6]. Sparse coding can model the interaction (inhibition) between the bases (neurons) by sparsifying their coefficients (activations), and our algorithms enable these phenomena to be tested with highly overcomplete bases.

First, we evaluated whether end-stopping behavior could be observed in the sparse coding framework. We generated random bars with different orientations and lengths in 14×14 image patches, and picked the stimulus bar which most strongly activates each basis, considering only the bases which are significantly activated by one of the test bars. For each such highly activated basis, and the corresponding optimal bar position and orientation, we vary length of the bar from 1 pixel to the maximal size and run sparse coding to measure the coefficients for the selected basis, relative to their maximum coefficient. As shown in Figure 3 (left), for highly overcomplete bases, we observe many cases in which the coefficient decreases significantly as the bar length is increased beyond the optimal point. This result is consistent with the end-stopping behavior of some V1 neurons.

Second, using the learned overcomplete bases, we tested for center-surround non-classical receptive field (nCRF) effects [7]. We found the optimal bar stimuli for 50 random bases and checked that these bases were among the most strongly activated ones for the optimal stimulus. For each of these

bases, we measured the response with its optimal bar stimulus with and without the aligned bar stimulus in the surround region (Figure 3 (right)). We then compared the basis response in these two cases to measure the suppression or facilitation due to the surround stimulus. The aligned surround stimuli produced a suppression of basis activation; 42 out of 50 bases showed suppression with aligned surround input images, and 13 bases among them showed more than 10% suppression, in qualitative accordance with observed nCRF surround suppression effects.

## 6 Application to self-taught learning

Sparse coding is an unsupervised algorithm that learns to represent input data succinctly using only a small number of bases. For example, using the "image edge" bases in Figure 2, it represents a new image patch $\vec{\xi}$ as a linear combination of just a small number of these bases $\vec{b}_j$. Informally, we think of this as finding a representation of an image patch in terms of the "edges" in the image; this gives a slightly higher-level/more abstract representation of the image than the pixel intensity values, and is useful for a variety of tasks.

In related work [8], we apply this to *self-taught learning*, a new machine learning formalism in which we are given a supervised learning problem together with additional unlabeled instances that may not have the same class labels as the labeled instances. For example, one may wish to learn to distinguish between cars and motorcycles given images of each, and additional—and in practice readily available—unlabeled images of various natural scenes. (This is in contrast to the much more restrictive semi-supervised learning problem, which would require that the unlabeled examples also be of cars or motorcycles only.) We apply our sparse coding algorithms to the unlabeled data to learn bases, which gives us a higher-level representation for images, thus making the supervised learning task easier. On a variety of problems including object recognition, audio classification, and text categorization, this approach leads to 11–36% reductions in test error.

## 7 Conclusion

In this paper, we formulated sparse coding as a combination of two convex optimization problems and presented efficient algorithms for each: the feature-sign search for solving the $L_1$-least squares problem to learn coefficients, and a Lagrange dual method for the $L_2$-constrained least squares problem to learn the bases for any sparsity penalty function. We test these algorithms on a variety of datasets, and show that they give significantly better performance compared to previous methods. Our algorithms can be used to learn an overcomplete set of bases, and show that sparse coding could partially explain the phenomena of end-stopping and nCRF surround suppression in V1 neurons.

**Acknowledgments.** We thank Bruno Olshausen, Pieter Abbeel, Sara Bolouki, Roger Grosse, Benjamin Packer, Austin Shoemaker and Joelle Skaf for helpful discussions. Support from the Office of Naval Research (ONR) under award number N00014-06-1-0828 is gratefully acknowledged.

## Footnotes

[1] We impose a norm constraint for bases: $\|\vec{b}_j\|^2 \leq c, \forall j = 1, ..., n$ for some constant $c$. Norm constraints are necessary because, otherwise, there always exists a linear transformation of $\vec{b}_j$'s and $\vec{s}^{(i)}$'s which keeps $\sum_{j=1}^{n} \vec{b}_j s_j^{(i)}$ unchanged, while making $s_j^{(i)}$'s approach zero. Based on similar motivation, Olshausen and Field used a scheme which retains the variation of coefficients for every basis at the same level [1, 2].

[2] A log (non-convex) penalty was used in [1]; thus, gradient-based methods can get stuck in local optima.

[3]A technical detail has been omitted from the algorithm for simplicity, as we have never observed it in practice. In Step 3 of the algorithm, in case $\hat{A}^\top\hat{A}$ becomes singular, we can check if $q \equiv \hat{A}^\top y - \gamma\hat{\theta}/2 \in \mathcal{R}(\hat{A}^\top\hat{A})$. If yes, we can replace the inverse with the pseudoinverse to minimize the unconstrained QP; otherwise, we can update $\hat{x}$ to the first zero-crossing along any direction $z$ such that $z \in \mathcal{N}(\hat{A}^\top\hat{A}), z^\top q \neq 0$. Both these steps are still guaranteed to reduce the objective; thus, the proof of convergence is unchanged.

[4]To simplify notation, we reuse $f(\cdot)$ even for subvectors such as $\hat{x}$; in the case of $f(\hat{x})$, we consider only the coefficients in $\hat{x}$ as variables, and all coefficients not in the subvector can be assumed constant at zero.

[5]If the algorithm terminates without reaching Step 2, we are done; otherwise, once the algorithm reaches Step 2, the same argument in the proof applies.

[6]The sparsity penalty for topographic cells can be written as $\sum_l \phi((\sum_{j \in \text{cell } l} s_j^2)^{\frac{1}{2}})$, where $\phi(\cdot)$ is a sparsity function and cell $l$ is a topographic cell (e.g., group of 'neighboring' bases in 2-D torus representation).

[7]We used the CVX package available at `http://www.stanford.edu/~boyd/cvx/`.

[8]LARS (with LASSO modification) provides the entire regularization path with discrete $L_1$-norm constraints; we further modified the algorithm so that it stops upon finding the optimal solution of the Equation (4).

[9]MATLAB code is available at `http://www-stat.stanford.edu/~atomizer/`.

[10]For each dataset/algorithm combination, we report the average running time over 20 trials.

[11]A general-purpose QP package (such as CVX) does not explicitly take the sparsity of the solutions into account. Thus, its solution tends to have many very small nonzero coefficients; as a result, the objective values obtained from CVX were always slightly worse than those obtained from feature-sign search or LARS.

[12]We ran each algorithm combination until the relative change of the objective per iteration became less than $10^{-6}$ (i.e., $|(f_{new} - f_{old})/f_{old}| < 10^{-6}$). To compute the running time to convergence, we first computed the "optimal" (minimum) objective value achieved by any algorithm combination. Then, for each combination, we defined the convergence point as the point at which the objective value reaches within 1% relative error of the observed "optimal" objective value. The running time measured is the time taken to reach this convergence point. We truncated the running time if the optimization did not converge within 60,000 seconds.

[13]We also evaluated a generic conjugate gradient implementation on the $L_1$ sparsity function; however, it did not converge even after 60,000 seconds.

[14]We used Lagrange dual formulation for learning bases, and both conjugate gradient with epsilon$L_1$ sparsity as well as the feature-sign search with $L_1$ sparsity for learning coefficients. The bases learned from both methods showed qualitatively similar receptive fields. The bases shown in the Figure 2 were learned using epsilon$L_1$ sparsity function and 4,000 input image patches randomly sampled for every iteration.

## References

[1] B. A. Olshausen and D. J. Field. Emergence of simple-cell receptive field properties by learning a sparse code for natural images. *Nature*, 381:607–609, 1996.

[2] B. A. Olshausen and D. J. Field. Sparse coding with an overcomplete basis set: A strategy employed by V1? *Vision Research*, 37:3311–3325, 1997.

[3] M. S. Lewicki and T. J. Sejnowski. Learning overcomplete representations. *Neural Comp.*, 12(2), 2000.

[4] B. A. Olshausen. Sparse coding of time-varying natural images. *Vision of Vision*, 2(7):130, 2002.

[5] B.A. Olshausen and D.J. Field. Sparse coding of sensory inputs. *Cur. Op. Neurobiology*, 14(4), 2004.

[6] M. P. Sceniak, M. J. Hawken, and R. Shapley. Visual spatial characterization of macaque V1 neurons. *The Journal of Neurophysiology*, 85(5):1873–1887, 2001.

[7] J.R. Cavanaugh, W. Bair, and J.A. Movshon. Nature and interaction of signals from the receptive field center and surround in macaque V1 neurons. *Journal of Neurophysiology*, 88(5):2530–2546, 2002.

[8] R. Raina, A. Battle, H. Lee, B. Packer, and A. Y. Ng. Self-taught learning. In *NIPS Workshop on Learning when test and training inputs have different distributions*, 2006.

[9] A. Y. Ng. Feature selection, $L_1$ vs. $L_2$ regularization, and rotational invariance. In *ICML*, 2004.

[10] Y. Censor and S. A. Zenios. *Parallel Optimization: Theory, Algorithms and Applications.* 1997.

[11] S. S. Chen, D. L. Donoho, and M. A. Saunders. Atomic decomposition by basis pursuit. *SIAM Journal on Scientific Computing*, 20(1):33–61, 1998.

[12] B. Efron, T. Hastie, I. Johnstone, and R. Tibshirani. Least angle regression. *Ann. Stat.*, 32(2), 2004.

[13] S. Perkins and J. Theiler. Online feature selection using grafting. In *ICML*, 2003.

[14] Aapo Hyvärinen, Patrik O. Hoyer, and Mika O. Inki. Topographic independent component analysis. *Neural Computation*, 13(7):1527–1558, 2001.

[15] J. H. van Hateren and A. van der Schaaf. Independent component filters of natural images compared with simple cells in primary visual cortex. *Proc.R.Soc.Lond. B*, 265:359–366, 1998.